# Kernel expansions with unlabeled examples

**Martin Szummer**
MIT AI Lab & CBCL
Cambridge, MA
szummer@ai.mit.edu

**Tommi Jaakkola**
MIT AI Lab
Cambridge, MA
tommi@ai.mit.edu

## Abstract

Modern classification applications necessitate supplementing the few available labeled examples with unlabeled examples to improve classification performance. We present a new tractable algorithm for exploiting unlabeled examples in discriminative classification. This is achieved essentially by expanding the input vectors into longer feature vectors via both labeled and unlabeled examples. The resulting classification method can be interpreted as a discriminative kernel density estimate and is readily trained via the EM algorithm, which in this case is both discriminative and achieves the optimal solution. We provide, in addition, a purely discriminative formulation of the estimation problem by appealing to the maximum entropy framework. We demonstrate that the proposed approach requires very few labeled examples for high classification accuracy.

## 1 Introduction

In many modern classification problems such as text categorization, very few labeled examples are available but a large number of unlabeled examples can be readily acquired. Various methods have recently been proposed to take advantage of unlabeled examples to improve classification performance. Such methods include the EM algorithm with naive Bayes models for text classification [1], the co-training framework [2], transduction [3, 4], and maximum entropy discrimination [5].

These approaches are divided primarily on the basis of whether they employ generative modeling or are motivated by robust classification. Unfortunately, the computational effort scales exponentially with the number of unlabeled examples for exact solutions in discriminative approaches such as transduction [3, 5]. Various approximations are available [4, 5] but their effect remains unclear.

In this paper, we formulate a complementary discriminative approach to exploiting unlabeled examples, effectively by using them to expand the representation of examples. This approach has several advantages including the ability to represent the true Bayes optimal decision boundary and making explicit use of the density over the examples. It is also computationally feasible as stated.

The paper is organized as follows. We start by discussing the kernel density estimate and providing a smoothness condition, assuming labeled data only. We subsequently introduce unlabeled data, define the expansion and formulate the EM algorithm for discriminative

training. In addition, we provide a purely discriminative version of the parameter estimation problem and formalize it as a maximum entropy discrimination problem. We then demonstrate experimentally that various concerns about the approach are not warranted.

## 2 Kernel density estimation and classification

We start by assuming a large number of *labeled* examples $D = \{(\mathbf{x}_1, \tilde{y}_1), \ldots, (\mathbf{x}_N, \tilde{y}_N)\}$, where $\tilde{y}_i \in \{-1, 1\}$ and $\mathbf{x}_i \in \mathcal{R}^d$. A joint kernel density estimate can be written as

$$P(\mathbf{x}, y) = \frac{1}{N} \sum_{i=1}^{N} \delta(y, \tilde{y}_i) \, K(\mathbf{x}, \mathbf{x}_i) \tag{1}$$

where $\int K(\mathbf{x}, \mathbf{x}_i) d\mu(\mathbf{x}) = 1$ for each $i$. With an appropriately chosen kernel $K$, a function of $N$, $P(\mathbf{x}, y)$ will be *consistent* in the sense of converging to the joint density as $N \to \infty$.

Given a fixed number of examples, the kernel functions $K(\mathbf{x}, \mathbf{x}_i)$ may be viewed as conditional probabilities $P(\mathbf{x}|i)$, where $i$ indexes the observed points. For the purposes of this paper, we assume a Gaussian form $K(\mathbf{x}, \mathbf{x}_i) = N(\mathbf{x}; \mathbf{x}_i, \sigma^2 I)$. The labels $\tilde{y}_i$ assigned to the sampled points $\mathbf{x}_i$ may themselves be noisy and we incorporate $P(y|i)$, a location-specific probability of labels. The resulting joint density model is

$$P(\mathbf{x}, y) = \frac{1}{N} \sum_{i=1}^{N} P(y|i) \, P(\mathbf{x}|i)$$

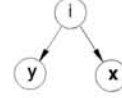

Interpreting $1/N$ as a prior probability of the index variable $i = 1, \ldots, N$, the resulting model conforms to the graph depicted above. This is reminiscent of the *aspect model* for clustering of dyadic data [6]. There are two main differences. First, the number of aspects here equals the number of examples and the model is not suitable for clustering. Second, we do not search for the probabilities $P(\mathbf{x}|i)$ (kernels), instead they are associated with each observed example and are merely adjusted in terms of scale (kernel width). This restriction yields a significant computational advantage in classification, which is the objective in this paper.

The posterior probability of the label $y$ given an example $\mathbf{x}$ is given by $P(y|\mathbf{x}) = \sum_i P(y|i)P(i|\mathbf{x})$, where $P(i|\mathbf{x}) \propto P(\mathbf{x}|i)/P(\mathbf{x})$ as $P(i)$ is assumed to be uniform. The quality of the posterior probability depends both on how accurately $P(y|i)$ are known as well as on the properties of the membership probabilities $P(i|\mathbf{x})$ (always known) that must be relatively smooth.

Here we provide a simple condition on the membership probabilities $P(i|\mathbf{x})$ so that any noise in the sampled labels for the available examples would not preclude accurate decisions. In other words, we wish to ensure that the conditional probabilities $P(y|\mathbf{x})$ can be evaluated accurately on the basis of the sampled estimate in Eq. (1). Removing the label noise provides an alternative way of setting the width parameter $\sigma$ of the Gaussian kernels. The simple lemma below, obtained via standard large deviation methods, ties the appropriate choice of the kernel width $\sigma$ to the squared norm of the membership probabilities $P(i|\mathbf{x}_j)$.

**Lemma 1** *Let $I_N = \{1, \ldots, N\}$. Given any $\delta > 0, \epsilon > 0$, and any collection of distributions $p_{i|k} \geq 0$, $\sum_{i \in I_N} p_{i|k} = 1$ for $k \in I_N$, such that $\|p_{\cdot|k}\|_2 \leq \epsilon/\sqrt{2 \log(2N/\delta)}, \forall k \in I_N$, and independent samples $\tilde{y}_i \in \{-1, 1\}$ from some $P(y|i), i \in I_N$, then $P(\exists k \in I_N : |\sum_{i=1}^{N} \tilde{y}_i p_{i|k} - \sum_{i=1}^{N} w_i p_{i|k}| > \epsilon) \leq \delta$ where $w_i = P(y = 1|i) - P(y = -1|i)$ and the probability is taken over the independent samples.*

The lemma applies to our case by setting $p_{i|k} = P(i|\mathbf{x}_k)$, $\{\tilde{y}_i\}$ represents the sampled labels for the examples, and by noting that the sign of $\sum w_i P(i|\mathbf{x})$ is the MAP decision rule from our model, $P(y = 1|\mathbf{x}) - P(y = -1|\mathbf{x})$. The lemma states that as long as the membership probabilities have appropriately bounded squared norm, the noise in the labeling is inconsequential for the classification decisions. Note, for example, that a distribution $p_{i|k} = 1/N$ has $\|p_{\cdot|k}\|_2 = 1/\sqrt{N}$ implying that the conditions are achievable for large $N$. The squared norm of $P(i|\mathbf{x})$ is directly controlled by the kernel width $\sigma^2$ and thus the lemma ties the kernel width with the accuracy of estimating the conditional probabilities $P(y|\mathbf{x})$. Algorithms for adjusting the kernel width(s) on the basis of this will be presented in a longer version of the paper.

## 3 The expansion and EM estimation

A useful way to view the resulting kernel density estimate is that each example $\mathbf{x}$ is represented by a vector of membership probabilities $P(i|\mathbf{x})$, $i = 1, \ldots, N$. Such *mixture distance* representations have been used extensively; it can also be viewed as a Fisher score vector computed with respect to adjustable weighting $P(i)$. The examples in this new representation are classified by associating $P(y|i)$ with each component and computing $P(y|\mathbf{x}) = \sum_i P(y|i)P(i|\mathbf{x})$. An alternative approach to exploiting kernel density estimates in classification is given by [7].

We now assume that we have labels for only a few examples, and our training data is $\{(\mathbf{x}_1, \tilde{y}_1), \ldots, (\mathbf{x}_L, \tilde{y}_L), \mathbf{x}_{L+1}, \ldots, \mathbf{x}_N\}$. In this case, we may continue to use the model defined above and estimate the free parameters, $P(y|i)$, $i = 1, \ldots, N$, from the few labeled examples. In other words, we can maximize the conditional log-likelihood

$$\sum_{l=1}^{L} \log P(\tilde{y}_l|\mathbf{x}_l) = \sum_{l=1}^{L} \log \sum_{i=1}^{N} P(\tilde{y}_l|i)P(i|\mathbf{x}_l) \tag{2}$$

where the first summation is only over the labeled examples and $L \ll N$. Since $P(i|\mathbf{x}_l)$ are fixed, this objective function is jointly concave in the free parameters and lends itself to a unique maximum value. The concavity also guarantees that this optimization is easily performed via the EM algorithm [8].

Let $p_{i|l}$ be the soft assignment for component $i$ given $(\mathbf{x}_l, \tilde{y}_l)$, i.e., $p_{i|l} = P(i|\mathbf{x}_l, \tilde{y}_l) \propto P(\tilde{y}_l|i)P(i|\mathbf{x}_l)$. The EM algorithm iterates between the E-step, where $p_{i|l}$ are recomputed from the current estimates of $P(y|i)$, and the M-step where we update $P(y|i) \leftarrow \sum_{l:\tilde{y}_l=y} p_{i|l} / \sum_l p_{i|l}$.

This procedure may have to be adjusted in cases where the overall frequency of different labels in the (labeled) training set deviates significantly from uniform. A simple rescaling $P(y|i) \leftarrow P(y|i)/L_y$ by the frequencies $L_y$ and renormalization after each M-step would probably suffice.

The runtime of this algorithm is $\mathcal{O}(LN)$. The discriminative formulation suggests that EM will provide reasonable parameter estimates $P(y|i)$ for classification purposes. The quality of the solution, as well as the potential for overfitting, is contingent on the smoothness of the kernels or, equivalently, smoothness of the membership probabilities $P(i|\mathbf{x})$. Note, however, that whether or not $P(y|i)$ will converge to the extreme values 0 or 1 is not an indication of overfitting. Actual classification decisions for unlabeled examples $\mathbf{x}_i$ (included in the expansion) need to be made on the basis of $P(y|\mathbf{x}_i)$ and not on the basis of $P(y|i)$, which function as parameters.

## 4 Discriminative estimation

An alternative discriminative formulation is also possible, one that is more sensitive to the decision boundary rather than probability values associated with the labels. To this end, consider the conditional probability $P(y|\mathbf{x}) = \sum_i P(y|i)P(i|\mathbf{x})$. The decisions are made on the basis of the sign of the discriminant function

$$f(\mathbf{x}) = P(y = 1|\mathbf{x}) - P(y = -1|\mathbf{x}) = \sum_{i=1}^{N} w_i P(i|\mathbf{x}) \tag{3}$$

where $w_i = P(y = 1|i) - P(y = -1|i)$. This is similar to a linear classifier and there are many ways of estimating the weights $w_i$ discriminatively. The weights should remain bounded, however, i.e., $w_i \in [-1, 1]$, so long as we wish to maintain the kernel density interpretation. Estimation algorithms with Euclidean norm regularization such as SVMs would not be appropriate in this sense. Instead, we employ the maximum entropy discrimination (MED) framework [5] and rely on the relation $w_i = E\{y_i\} = \sum_{y_i=\pm 1} y_i P(y)$ to estimate the distribution $P(y)$ over all the labels $y = [y_1, \ldots, y_N]$. Here $y_i$ is a parameter associated with the $i^{th}$ example and should be distinguished from any observed labels. We can show that in this case the maximum entropy solution factors across the examples $P(y_1, \ldots, y_N) = \prod_i P_i(y_i)$ and we can formulate the estimation problem directly in terms of the marginals $P_i(y_i)$.

The maximum entropy formalism encodes the principle that label assignments $P_i(y_i)$ for the examples should remain uninformative to the extent possible given the classification objective. More formally, given a set of $L$ labeled examples $(\mathbf{x}_1, \tilde{y}_1), \ldots, (\mathbf{x}_L, \tilde{y}_L)$, we maximize $\sum_{i=1}^{N} H(y_i) - C \sum_l \xi_l$ subject to the classification constraints

$$\tilde{y}_l \left[ \sum_{i=1}^{N} \sum_{y_i=\pm 1} y_i P_i(y_i) P(i|\mathbf{x}_l) \right] + \xi_l \geq \gamma \quad \forall l \in [1 \ldots L] \tag{4}$$

where $H(y_i)$ is the entropy of $y_i$ relative to the marginal $P_i(y_i)$. Here $\gamma$ specifies the target separation ($\gamma \in [0, 1]$) and the slack variables $\xi_l \geq 0$ permit deviations from the target to ensure that a solution always exists. The solution is not very sensitive to these parameters, and $\gamma = 0.1$ and $C = 40$ worked well for many problems. The advantage of this formulation is that effort is spent only on those training examples whose classification is uncertain. Examples already classified correctly with a margin larger than $\gamma$ are effectively ignored. The optimization problem and algorithms are explained in the appendix.

## 5 Discussion of the expanded representation

The kernel expansion enables us to *represent* the Bayes optimal decision boundary provided that the kernel density estimate is sufficiently accurate. With this representation, the EM and MED algorithms actually estimate decision boundaries that are sensitive to the density $P(\mathbf{x})$. For example, labeled points in high-density regions will influence the boundary more than in low-density regions. The boundary will partly follow the density, but unlike in unsupervised methods, will adhere strongly to the labeled points. Moreover, our estimation techniques limit the effect of outliers, as all points have a bounded weight $w_i = [-1, 1]$ (spurious unlabeled points do not adversely affect the boundary).

As we impose smoothness constraints on the membership probabilities $P(i|\mathbf{x})$, we also guarantee that the capacity of the resulting classifier need not increase with the number of unlabeled examples (in the fat shattering sense). Also, in the context of the maximum entropy formulation, if a point is not helpful for the classification constraints, then entropy

is maximized for $P_i(y = \pm 1) = 0.5$, implying $w_i = 0$, and the point has no effect on the boundary.

If we dispense with the conditional probability interpretation of the kernels $K$, we are free to choose them from a more general class of functions. For example, the kernels no longer have to integrate to 1. An expansion of $\mathbf{x}$ in terms of these kernels can still be meaningful; as a special case, when linear kernels are chosen, the expansion reduces to weighting distances between points by the covariance of the data. Distinctions along high variance directions then become easier to make, which is helpful when between-class scatter is greater than within-class scatter.

Thus, even though the probabilistic interpretation is missing, a simple preprocessing step can still help, e.g., support vector machines to take advantage of unlabeled data: we can expand the inputs $\mathbf{x}$ in terms of kernels $G$ from labeled and unlabeled points as in $\phi(\mathbf{x}) = \frac{1}{Z}[G(\mathbf{x}, \mathbf{x}_1), \ldots, G(\mathbf{x}, \mathbf{x}_N)]$, where $Z$ optionally normalizes the feature vector.

## 6 Results

We first address the potential concern that the expanded representation may involve too many degrees of freedom and result in poor generalization. Figure 1a) demonstrates that this is not the case and, instead, the test classification error approaches the limiting asymptotic rate exponentially fast. The problem considered was a DNA splice site classification problem with 500 examples for which $d = 100$. Varying sizes of random subsets were labeled and all the examples were used in the expansion as unlabeled examples. The error rate was computed on the basis of the remaining $500 - L$ examples without labels, where $L$ denotes the number of labeled examples. The results in the figure were averaged across 20 independent runs. The exponential rate of convergence towards the limiting rate is evidenced by the linear trend in the semilog figure 1a). The mean test errors shown in figure 1b) indicate that the purely discriminative training (MED) can contribute substantially to the accuracy. The kernel width in these experiments was simply fixed to the median distance to the 5th nearest neighbor from the opposite class. Results from other methods of choosing the kernel width (the squared norm, adaptive) will be discussed in the longer version of the paper.

Another concern is perhaps that the formulation is valid only in cases where we have a large number of unlabeled examples. In principle, the method could deteriorate rapidly after the kernel density estimate no longer can be assumed to give reasonable estimates. Figure 2a) illustrates that this is not a valid interpretation. The problem here is to classify DNA microarray experiments on the basis of the leukemia types that the tissues used in the array experiments corresponded to. Each input vector for the classifier consists of the expression levels of over 7000 genes that were included as probes in the arrays. The number of examples available was 38 for training and 34 for testing. We included all examples as unlabeled points in the expansion and randomly selected subsets of labeled training examples, and measured the performance only on the test examples (which were of slightly different type and hence more appropriate for assessing generalization). Figure 2 shows rapid convergence for EM and the discriminative MED formulation. The "asymptotic" level here corresponds to about one classification error among the 34 test examples. The results were averaged over 20 independent runs.

## 7 Conclusion

We have provided a complementary framework for exploiting unlabeled examples in discriminative classification problems. The framework involves a combination of the ideas of kernel density estimation and representational expansion of input vectors. A simple EM

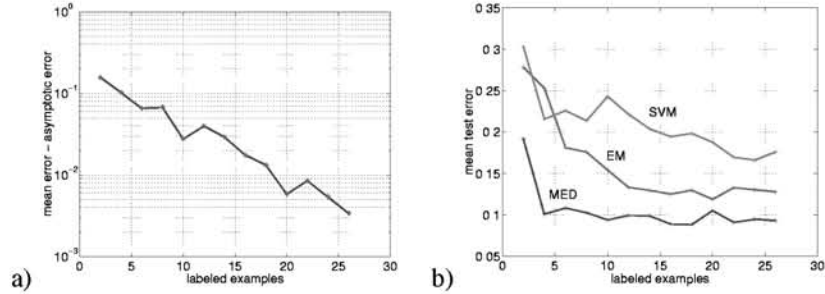

a)    b)

Figure 1: A semilog plot of the test error rate for the EM formulation less the asymptotic rate as a function of labeled examples. The linear trend in the figure implies that the error rate approaches the asymptotic error exponentially fast. b) The mean test errors for EM, MED and SVM as a function of the number of labeled examples. SVM does not use unlabeled examples.

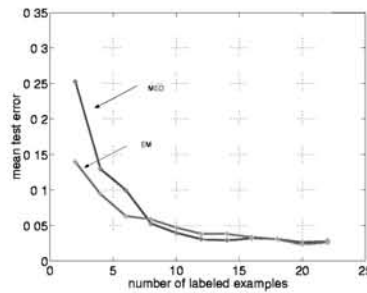

Figure 2: The mean test errors for the leukemia classification problem as a function of the number of randomly chosen labeled examples. Results are given for both EM (lower line) and MED (upper line) formulations.

algorithm is sufficient for finding globally optimal parameter estimates but we have shown that a purely discriminative formulation can yield substantially better results within the framework.

Possible extensions include using the kernel expansions with transductive algorithms that enforce margin constraints also for the unlabeled examples [5]. Such combination can be particularly helpful in terms of capturing the lower dimensional structure of the data. Other extensions include analysis of the framework similarly to [9].

## Acknowledgments

The authors gratefully acknowledge support from NTT and NSF. Szummer would also like to thank Thomas Minka for many helpful discussions and insights.

## References

[1] Nigam K., McCallum A., Thrun S., and Mitchell T. (2000) Text classification from labeled and unlabeled examples. *Machine Learning* **39** (2):103–134.

[2] Blum A., Mitchell T. (1998) Combining Labeled and Unlabeled Data with Co-Training. In *Proc. 11th Annual Conf. Computational Learning Theory*, pp. 92–100.

[3] Vapnik V. (1998) *Statistical learning theory*. John Wiley & Sons.

[4] Joachims, T. (1999) Transductive inference for text classification using support vector machines. *International Conference on Machine Learning*.

[5] Jaakkola T., Meila M., and Jebara T. (1999) Maximum entropy discrimination. In *Advances in Neural Information Processing Systems 12*.

[6] Hofmann T., Puzicha J. (1998) Unsupervised Learning from Dyadic Data. International Computer Science Institute, TR-98-042.

[7] Tong S., Koller D. (2000) Restricted Bayes Optimal Classifiers. *Proceedings AAAI*.

[8] Miller D., Uyar T. (1996) A Mixture of Experts Classifer with Learning Based on Both Labelled and Unlabelled Data. In *Advances in Neural Information Processing Systems 9*, pp. 571–577.

[9] Castelli V., Cover T. (1996) The relative value of labeled and unlabeled samples in pattern recognition with an unknown mixing parameter. *IEEE Transactions on information theory* **42** (6): 2102–2117.

## A Maximum entropy solution

The unique solution to the maximum entropy estimation problem is found via introducing Lagrange multipliers $\{\lambda_l\}$ for the classification constraints. The multipliers satisfy $\lambda_l \in [0, C]$, where the lower bound comes from the inequality constraints and the upper bound from the linear margin penalties being minimized. To represent the solution and find the optimal setting of $\lambda_l$ we must evaluate the partition function

$$Z(\lambda) = e^{-\sum_l^L \lambda_l \gamma} \sum_{y_1,\ldots,y_N} \prod_{i=1}^N e^{\sum_l^L \tilde{y}_l \lambda_l y_i P(i|\mathbf{x}_l)} = \tag{5}$$

$$= e^{-\sum_l^L \lambda_l \gamma} \prod_{i=1}^N \left( e^{\sum_l^L \tilde{y}_l \lambda_l P(i|\mathbf{x}_l)} + e^{-\sum_l^L \tilde{y}_l \lambda_l P(i|\mathbf{x}_l)} \right) \tag{6}$$

that normalizes the maximum entropy distribution. Here $\tilde{y}$ denote the observed labels. Minimizing the jointly convex log-partition function $\log Z(\lambda)$ with respect to the Lagrange multipliers leads to the optimal setting $\{\lambda_l^*\}$. This optimization is readily done via an axis parallel line search (e.g. the bisection method). The required gradients are given by

$$\frac{\partial \log Z(\lambda)}{\partial \lambda_k} = -\gamma + \sum_{i=1}^N \tanh\left( \sum_{l=1}^L \tilde{y}_l \lambda_l^* P(i|\mathbf{x}_l) \right) \tilde{y}_k P(i|\mathbf{x}_k) = \tag{7}$$

$$= -\gamma + \tilde{y}_k \sum_{i=1}^N E_{P_i^*}\{y_i\} P(i|\mathbf{x}_k) \tag{8}$$

(this is essentially the classification constraint). The expectation is taken with respect to the maximum entropy distribution $P^*(y_1, \ldots, y_N) = P_1^*(y_1) \cdots P_N^*(y_N)$ where the components are $P_i^*(y_i) \propto \exp\{\sum_l \tilde{y}_l \lambda_l y_i P(i|\mathbf{x})\}$. The label averages $w_i^* = E_{P^*}\{y_i\} = \sum_{y_i} y_i P_i^*(y_i)$ are needed for the decision rule as well as in the optimization. We can identify these from above $w_i^* = \tanh(\sum_l \tilde{y}_l \lambda_l^* P(i|\mathbf{x}_l))$ and they are readily evaluated. Finding the solution involves $\mathcal{O}(L^2 N)$ operations.

Often the numbers of positive and negative training labels are imbalanced. The MED formulation (analogously to SVMs) can be adjusted by defining the margin penalties as $C^+ \sum_{l:\tilde{y}_l=1} \xi_l + C^- \sum_{l:\tilde{y}_l=-1} \xi_l$, where, for example, $L^+ C^+ = L^- C^-$ that equalizes the mean penalties. The coefficients $C^+$ and $C^-$ can also be modified adaptively during the estimation process to balance the rate of misclassification errors across the two classes.